# PAC-Bayesian Analysis of Contextual Bandits

**Yevgeny Seldin**[1,4]  **Peter Auer**[2]  **François Laviolette**[3]  **John Shawe-Taylor**[4]  **Ronald Ortner**[2]
[1]Max Planck Institute for Intelligent Systems, Tübingen, Germany
[2]Chair for Information Technology, Montanuniversität Leoben, Austria
[3]Département d'informatique, Université Laval, Québec, Canada
[4]Department of Computer Science, University College London, UK
seldin@tuebingen.mpg.de, {auer,ronald.ortner}@unileoben.ac.at,
francois.laviolette@ift.ulaval.ca, jst@cs.ucl.ac.uk

## Abstract

We derive an instantaneous (per-round) data-dependent regret bound for stochastic multiarmed bandits with side information (also known as contextual bandits). The scaling of our regret bound with the number of states (contexts) $N$ goes as $\sqrt{N I_{\rho_t}(S; A)}$, where $I_{\rho_t}(S; A)$ is the mutual information between states and actions (the side information) used by the algorithm at round $t$. If the algorithm uses all the side information, the regret bound scales as $\sqrt{N \ln K}$, where $K$ is the number of actions (arms). However, if the side information $I_{\rho_t}(S; A)$ is not fully used, the regret bound is significantly tighter. In the extreme case, when $I_{\rho_t}(S; A) = 0$, the dependence on the number of states reduces from linear to logarithmic. Our analysis allows to provide the algorithm large amount of side information, let the algorithm to decide which side information is relevant for the task, and penalize the algorithm only for the side information that it is using de facto. We also present an algorithm for multiarmed bandits with side information with $O(K)$ computational complexity per game round.

## 1 Introduction

Multiarmed bandits with side information are an elegant mathematical model for many real-life interactive systems, such as personalized online advertising, personalized medical treatment, and so on. This model is also known as contextual bandits or associative bandits (Kaelbling, 1994, Strehl et al., 2006, Langford and Zhang, 2007, Beygelzimer et al., 2011). In multiarmed bandits with side information the learner repeatedly observes states (side information) $\{s_1, s_2, \dots\}$ (for example, symptoms of a patient) and has to perform actions (for example, prescribe drugs), such that the expected regret is minimized. The regret is usually measured by the difference between the reward that could be achieved by the best (unknown) fixed policy (for example, the number of patients that would be cured if we knew the best drug for each set of symptoms) and the reward obtained by the algorithm (the number of patients that were actually cured).

Most of the existing analyses of multiarmed bandits with side information has focused on the adversarial (worst-case) model, where the sequence of rewards associated with each state-action pair is chosen by an adversary. However, many problems in real-life are not adversarial. We derive data-dependent analysis for stochastic multiarmed bandits with side information. In the stochastic setting the rewards for each state-action pair are drawn from a fixed unknown distribution. The sequence of states is also drawn from a fixed unknown distribution. We restrict ourselves to problems with finite number of states $N$ and finite number of actions $K$ and leave generalization to continuous state and action spaces to future work. We also do not assume any structure of the state space. Thus, for us a state is just a number between 1 and $N$. For example, in online advertising the state can be the country from which a web page is accessed.

The result presented in this paper exhibits adaptive dependency on the side information (state identity) that is actually used by the algorithm. This allows us to provide the algorithm a large amount of side information and let the algorithm decide, which of this side information is actually relevant to the task. For example, in online advertising we can increase the state resolution and provide the algorithm the town from which the web page was accessed, but if this refined state information is not used by the algorithm the regret bound will not deteriorate. This can be opposed to existing analysis of adversarial multiarmed bandits, where the regret bound depends on a predefined complexity of the underlying expert class (Beygelzimer et al., 2011). Thus, the existing analysis of adversarial multiarmed bandits would either become looser if we add more side information or a-priori limit the usage of the side information through its internal structure. (We note that through the relation between PAC-Bayesian analysis and the analysis of adversarial online learning described in Banerjee (2006) it might be possible to extend our analysis to adversarial setting, but we leave this research direction to future work.)

The idea of regularization by relevant mutual information goes back to the Information Bottleneck principle in supervised and unsupervised learning (Tishby et al., 1999). Tishby and Polani (2010) further suggested to measure the complexity of a policy in reinforcement learning by the mutual information between states and actions used by the policy. We note, however, that our starting point is the regret bound and we derive the regularization term from our analysis without introducing it a-priori. The analysis also provides time and data dependent weighting of the regularization term.

Our results are based on PAC-Bayesian analysis (Shawe-Taylor and Williamson, 1997, Shawe-Taylor et al., 1998, McAllester, 1998, Seeger, 2002), which was developed for supervised learning within the PAC (Probably Approximately Correct) learning framework (Valiant, 1984). In PAC-Bayesian analysis the complexity of a model is defined by a user-selected prior over a hypothesis space. Unlike in VC-dimension-based approaches and their successors, where the complexity is defined for a hypothesis class, in PAC-Bayesian analysis the complexity is defined for individual hypotheses. The analysis provides an explicit trade-off between individual model complexity and its empirical performance and a high probability guarantee on the expected performance.

An important distinction between supervised learning and problems with limited feedback, such as multiarmed bandits and reinforcement learning more generally, is the fact that in supervised learning the training set is given, whereas in reinforcement learning the training set is generated by the learner as it plays the game. In supervised learning every hypothesis in a hypothesis class can be evaluated on all the samples, whereas in reinforcement learning rewards of one action cannot be used to evaluate another action. Recently, Seldin et al. (2011b,a) generalized PAC-Bayesian analysis to martingales and suggested a way to apply it under limited feedback. Here, we apply this generalization to multiarmed bandits with side information.

The remainder of the paper is organized as follows. We start with definitions in Section 2 and provide our main results in Section 3, which include an instantaneous regret bound and a new algorithm for stochastic multiarmed bandits with side information. In Section 4 we present an experiment that illustrates our theoretical results. Then, we dive into the proof of our main results in Section 5 and discuss the paper in Section 6.

## 2   Definitions

In this section we provide all essential definitions for our main results in the following section. We start with the definition of stochastic multiarmed bandits with side information. Let $\mathcal{S}$ be a set of $|\mathcal{S}| = N$ states and let $\mathcal{A}$ be a set of $|\mathcal{A}| = K$ actions, such that any action can be performed in any state. Let $s \in \mathcal{S}$ denote the states and $a \in \mathcal{A}$ denote the actions. Let $R(a, s)$ be the expected reward for performing action $a$ in state $s$. At each round $t$ of the game the learner is presented a state $S_t$ drawn i.i.d. according to an unknown distribution $p(s)$. The learner draws an action $A_t$ according to his choice of a distribution (policy) $\pi_t(a|s)$ and obtains a stochastic reward $R_t$ with expected value $R(A_t, S_t)$. Let $\{S_1, S_2, \dots\}$ denote the sequence of observed states, $\{\pi_1, \pi_2, \dots\}$ the sequence of policies played, $\{A_1, A_2, \dots\}$ the sequence of actions played, and $\{R_1, R_2, \dots\}$ the sequence of observed rewards. Let $\mathcal{T}_t = \{\{S_1, \dots, S_t\}, \{\pi_1, \dots, \pi_t\}, \{A_1, \dots, A_t\}, \{R_1, \dots, R_t\}\}$ denote the history of the game up to time $t$.

Assume that $\pi_t(a|s) > 0$ for all $t$, $a$, and $s$. For $t \geq 1$, $a \in \{1, \dots, K\}$, and the sequence of observed states $\{S_1, \dots, S_t\}$ define a set of random variables $R_t^{a,S_t}$:

$$R_t^{a,S_t} = \begin{cases} \frac{1}{\pi_t(a|S_t)} R_t, & \text{if } A_t = a \\ 0, & \text{otherwise.} \end{cases}$$

(The variables $R_t^{a,s}$ are defined only for the observed state $s = S_t$.) Note that whenever defined, $\mathbb{E}[R_t^{a,S_t}|\mathcal{T}_{t-1}, S_t] = R(a, S_t)$. The definition of $R_t^{a,s}$ is generally known as importance weighted sampling (Sutton and Barto, 1998). Importance weighted sampling is required for application of PAC-Bayesian analysis, as will be shown in the technical part of the paper.

Define $n_t(s) = \sum_{\tau=1}^{t} \mathbb{I}_{\{S_\tau = s\}}$ as the number of times state $s$ appeared up to time $t$ ($\mathbb{I}$ is the indicator function). We define the empirical rewards of state-action pairs as:

$$\hat{R}_t(a, s) = \begin{cases} \frac{\sum_{\{\tau=1,\dots,t:S_\tau=s\}} R_\tau^{a,s}}{n_t(s)}, & \text{if } n_t(s) > 0 \\ 0, & \text{otherwise.} \end{cases}$$

Note that whenever $n_t(s) > 0$ we have $\mathbb{E}\hat{R}_t(a, s) = R(a, s)$. For every state $s$ we define the "best" action in that state as $a^* = \arg\max_a R(a, s)$ (if there are multiple "best" actions, one of them is chosen arbitrarily). We then define the expected and empirical regret for performing any other action $a$ in state $s$ as:

$$\Delta(a, s) = R(a^*(s), s) - R(a, s), \qquad \hat{\Delta}_t(a, s) = \hat{R}_t(a^*(s), s) - \hat{R}_t(a, s).$$

Let $\hat{p}_t(s) = \frac{n_t(s)}{t}$ be the empirical distribution over states observed up to time $t$. For any policy $\rho(a|s)$ we define the empirical reward, empirical regret, and expected regret of the policy as: $\hat{R}_t(\rho) = \sum_s \hat{p}_t(s) \sum_a \rho(a|s)\hat{R}_t(a, s)$, $\hat{\Delta}_t(\rho) = \sum_s \hat{p}_t(s) \sum_a \rho(a|s)\hat{\Delta}_t(a, s)$, and $\Delta(\rho) = \sum_s p(s) \sum_a \rho(a|s)\Delta(a, s)$.

We define the marginal distribution over actions that corresponds to a policy $\rho(a|s)$ and the uniform distribution over $\mathcal{S}$ as $\bar{\rho}(a) = \frac{1}{N} \sum_s \rho(a|s)$ and the mutual information between actions and states corresponding to the policy $\rho(a|s)$ and the uniform distribution over $\mathcal{S}$ as

$$I_\rho(S; A) = \frac{1}{N} \sum_{s,a} \rho(a|s) \ln \frac{\rho(a|s)}{\bar{\rho}(a)}.$$

For the proof of our main result and also in order to explain the experiments we also have to define a hypothesis space for our problem. This definition is not used in the statement of the main result. Let $\mathcal{H}$ be a hypothesis space, such that each member $h \in \mathcal{H}$ is a deterministic mapping from $\mathcal{S}$ to $\mathcal{A}$. Denote by $a = h(s)$ the action assigned by hypothesis $h$ to state $s$. It is easy to see that the size of the hypothesis space $|\mathcal{H}| = K^N$. Denote by $R(h) = \sum_{s \in \mathcal{S}} p(s)R(h(s), s)$ the expected reward of a hypothesis $h$. Define:

$$\hat{R}_t(h) = \frac{1}{t} \sum_{\tau=1}^{t} R_\tau^{h(S_\tau), S_\tau}.$$

Note that $\mathbb{E}\hat{R}_t(h) = R(h)$.

Let $h^* = \arg\max_{h \in \mathcal{H}} R(h)$ be the "best" hypothesis (the one that chooses the "best" action in each state). (If there are multiple hypotheses achieving maximal reward pick any of them.) Define:

$$\Delta(h) = R(h^*) - R(h), \qquad \hat{\Delta}_t(h) = \hat{R}_t(h^*) - \hat{R}_t(h).$$

Any policy $\rho(a|s)$ defines a distribution over $\mathcal{H}$: we can draw an action $a$ for each state $s$ according to $\rho(a|s)$ and thus obtain a hypothesis $h \in \mathcal{H}$. We use $\rho(h)$ to denote the respective probability of drawing $h$. For a policy $\rho$ we define $\Delta(\rho) = \mathbb{E}_{\rho(h)}[\Delta(h)]$ and $\hat{\Delta}_t(\rho) = \mathbb{E}_{\rho(h)}[\hat{\Delta}_t(h)]$. By marginalization these definitions are consistent with our preceding definitions of $\Delta(\rho)$ and $\hat{\Delta}_t(\rho)$.

Finally, let $n_h(a) = \sum_{s=1}^{N} \mathbb{I}_{h(s)=a}$ be the number of states in which action $a$ is played by the hypothesis $h$. Let $A^h = \left\{ \frac{n_h(a)}{N} \right\}_{a \in \mathcal{A}}$ be the normalized cardinality profile (histogram) over the

actions played by hypothesis $h$ (with respect to the uniform distribution over $\mathcal{S}$). Let $H(A^h) = -\sum_a \frac{n_h(a)}{N} \ln \frac{n_h(a)}{N}$ be the entropy of this cardinality profile. In other words, $H(A^h)$ is the entropy of an action choice of hypothesis $h$ (with respect to the uniform distribution over $\mathcal{S}$). Note, that the optimal policy $\rho^*(a|s)$ (the one, that selects the "best" action in each state) is deterministic and we have $I_{\rho^*}(S;A) = H(A^{h^*})$.

## 3  Main Results

Our main result is a data and complexity dependent regret bound for a general class of prediction strategies of a smoothed exponential form. Let $\rho_t(a)$ be an arbitrary distribution over actions, let

$$\rho_t^{exp}(a|s) = \frac{\rho_t(a)e^{\gamma_t \hat{R}_t(a,s)}}{Z(\rho_t^{exp}, s)}, \tag{1}$$

where $Z(\rho_t^{exp}, s) = \sum_a \rho_t(a)e^{\gamma_t \hat{R}_t(a,s)}$ is a normalization factor, and let

$$\tilde{\rho}_t^{exp}(a|s) = (1 - K\varepsilon_{t+1})\rho_t^{exp}(a|s) + \varepsilon_{t+1} \tag{2}$$

be a smoothed exponential policy. The following theorem provides a regret bound for playing $\tilde{\rho}_t^{exp}$ at round $t+1$ of the game. For generality, we assume that rounds $1, \ldots, t$ were played according to arbitrary policies $\pi_1, \ldots, \pi_t$.

**Theorem 1.** *Assume that in game rounds $1, \ldots, t$ policies $\{\pi_1, \ldots, \pi_t\}$ were played and assume that $\min_{a,s} \pi_t(a|s) \geq \varepsilon_t$ for an arbitrary $\varepsilon_t$ that is independent of $\mathcal{T}_t$. Let $\rho_t(a)$ be an arbitrary distribution over $\mathcal{A}$ that can depend on $\mathcal{T}_t$ and satisfies $\min_a \rho_t(a) \geq \epsilon_t$. Let $c > 1$ be an arbitrary number that is independent of $\mathcal{T}_t$. Then, with probability greater than $1 - \delta$ over $\mathcal{T}_t$, simultaneously for all policies $\tilde{\rho}_t^{exp}$ defined by (2) that satisfy*

$$\frac{NI_{\rho_t^{exp}}(S;A) + K(\ln N + \ln K) + \ln \frac{2m_t}{\delta}}{2(e-2)t} \leq \frac{\varepsilon_t}{c^2} \tag{3}$$

*we have*:

$$\Delta(\tilde{\rho}_t^{exp}) \leq (1+c)\sqrt{\frac{2(e-2)(NI_{\rho_t^{exp}}(S;A) + K(\ln N + \ln K) + \ln \frac{2m_t}{\delta})}{t\varepsilon_t}} + \frac{\ln \frac{1}{\epsilon_{t+1}}}{\gamma_t} + K\varepsilon_{t+1}, \tag{4}$$

*where $m_t = \ln\left(\sqrt{\frac{(e-2)t}{\ln \frac{2}{\delta}}}\right) / \ln(c)$, and for all $\rho_t^{exp}$ that do not satisfy (3), with the same probability*:

$$\Delta(\tilde{\rho}_t^{exp}) \leq \frac{2(NI_{\rho_t^{exp}}(S;A) + K(\ln N + \ln K) + \ln \frac{2m_t}{\delta})}{t\varepsilon_t} + \frac{\ln \frac{1}{\epsilon_{t+1}}}{\gamma_t} + K\varepsilon_{t+1}.$$

Note that the mutual information in Theorem 1 is calculated with respect to $\rho_t^{exp}$ and not $\tilde{\rho}_t^{exp}$. Theorem 1 allows to tune the learning rate $\gamma_t$ based on the sample. It also provides an instantaneous regret bound for any algorithm that plays the policies $\{\tilde{\rho}_1^{exp}, \tilde{\rho}_2^{exp}, \ldots\}$ throughout the game. In order to obtain such a bound we just have to take a decreasing sequence $\{\varepsilon_1, \varepsilon_2, \ldots\}$ and substitute $\delta$ in Theorem 1 with $\delta_t = \frac{\delta}{t(t+1)}$. Then, by the union bound, the result holds with probability greater than $1 - \delta$ for all rounds of the game simultaneously. This leads to Algorithm 1 for stochastic multiarmed bandits with side information. Note that each round of the algorithm takes $O(K)$ time.

Theorem 1 is based on the following regret decomposition and the subsequent theorem and two lemmas that bound the three terms in the decomposition.

$$\Delta(\tilde{\rho}_t^{exp}) = [\Delta(\rho_t^{exp}) - \hat{\Delta}_t(\rho_t^{exp})] + \hat{\Delta}_t(\rho_t^{exp}) + [R(\rho_t^{exp}) - R(\tilde{\rho}_t^{exp})]. \tag{5}$$

**Theorem 2.** *Under the conditions of Theorem 1 on $\{\pi_1, \ldots, \pi_t\}$ and $c$, simultaneously for all policies $\rho$ that satisfy (3) with probability greater than $1 - \delta$*:

$$\left|\Delta(\rho) - \hat{\Delta}_t(\rho)\right| \leq (1+c)\sqrt{\frac{2(e-2)(NI_{\rho(S;A)} + K(\ln N + \ln K) + \ln \frac{2m_t}{\delta})}{t\varepsilon_t}}, \tag{6}$$

**Algorithm 1:** Algorithm for stochastic contextual bandits. (See text for definitions of $\varepsilon_t$ and $\gamma_t$.)

---

**Input**: N, K
$\hat{R}(a,s) \leftarrow 0$ for all $a,s$ (These are cumulative [unnormalized] rewards)
$\rho(a) \leftarrow \frac{1}{K}$ for all $a$
$n(s) \leftarrow 0$ for all $s$
$t \leftarrow 1$
**while** *not terminated* **do**

    Observe state $S_t$.
    **if** $\left(\varepsilon_t \geq \frac{1}{K}\right)$ *or* $(n(S_t) = 0)$ **then**
        $\rho(a|S_t) \leftarrow \rho(a)$ for all $a$
    **else**
        $\rho(a|S_t) \leftarrow (1 - K\varepsilon_t)\frac{\rho(a)e^{\gamma_t \hat{R}(a,S_t)/n(S_t)}}{\sum_{a'}\rho(a')e^{\gamma_t \hat{R}(a',S_t)/n(S_t)}} + \varepsilon_t$ for all $a$
        $\rho(a) \leftarrow \frac{N-1}{N}\rho(a) + \frac{1}{N}\rho(a|S_t)$ for all $a$
    Draw action $A_t$ according to $\rho(a|S_t)$ and play it.
    Observe reward $R_t$.
    $n(S_t) \leftarrow n(S_t) + 1$
    $\hat{R}(A_t, S_t) \leftarrow \hat{R}(A_t, S_t) + \frac{R_t}{\rho(A_t|S_t)}$
    $t \leftarrow t + 1$

---

*and for all $\rho$ that do not satisfy* (3) *with the same probability*:

$$\left|\Delta(\rho) - \hat{\Delta}_t(\rho)\right| \leq \frac{2(NI_\rho(S;A) + K(\ln N + \ln K) + \ln\frac{2m_t}{\delta})}{t\varepsilon_t}.$$

Note that Theorem 2 holds for all possible $\rho$-s, including those that do not have an exponential form.

**Lemma 1.** *For any distribution $\rho_t^{exp}$ of the form* (1)*, where $\rho_t(a) \geq \epsilon$ for all $a$, we have*:

$$\hat{\Delta}_t(\rho_t^{exp}) \leq \frac{\ln\frac{1}{\epsilon}}{\gamma_t}.$$

**Lemma 2.** *Let $\tilde{\rho}$ be an $\varepsilon$-smoothed version of a policy $\rho$, such that $\tilde{\rho}(a|s) = (1 - K\varepsilon)\rho(a|s) + \varepsilon$, then*

$$R(\rho) - R(\tilde{\rho}) \leq K\varepsilon.$$

Proof of Theorem 2 is provided in Section 5 and proofs of Lemmas 1 and 2 are provided in the supplementary material.

**Comments on Theorem 1.** Theorem 1 exhibits what we were looking for: the regret of a policy $\tilde{\rho}_t^{exp}$ depends on the trade-off between its complexity, $NI_{\rho_t^{exp}}(S;A)$, and the empirical regret, which is bounded by $\frac{1}{\gamma_t}\ln\frac{1}{\epsilon_{t+1}}$. We note that $0 \leq I_{\rho_t}(S;A) \leq \ln K$, hence, the result is interesting when $N \gg K$, since otherwise $K\ln K$ term in the bound neutralizes the advantage we get from having small mutual information values. The assumption that $N \gg K$ is reasonable for many applications.

We believe that the dependence of the first term of the regret bound (4) on $\varepsilon_t$ is an artifact of our crude upper bound on the variance of the sampling process (given in Lemma 3 in the proof of Theorem 2) and that this term should not be in the bound. This is supported by an empirical study of stochastic multiarmed bandits (Seldin et al., 2011a). With the current bound the best choice for $\varepsilon_t$ is $\varepsilon_t = (Kt)^{-1/3}$, which, by integration over the game rounds, yields $O(K^{1/3}t^{2/3})$ dependence of the cumulative regret on the number of arms and game rounds. However, if we manage to derive a tighter analysis and remove $\varepsilon_t$ from the first term in (4), the best choice of $\varepsilon_t$ will be $\varepsilon_t = (Kt)^{-1/2}$ and the dependence of the cumulative regret on the number of arms and time horizon will improve to $O((Kt)^{1/2})$. One way to achieve this is to apply EXP3.P-style updates (Auer et al., 2002b), however, Seldin et al. (2011a) empirically show that in stochastic environments EXP3 algorithm of Auer et al. (2002b), which is closely related to Algorithm 1, has significantly better performance. Thus, it is desirable to derive a better analysis for EXP3 algorithm in stochastic environments. We note

that although UCB algorithm for stochastic multiarmed bandits (Auer et al., 2002a) is asymptotically better than the EXP3 algorithm, it is not compatible with PAC-Bayesian analysis and we are not aware of a way to derive a UCB-type algorithm and analysis for multiarmed bandits with side information, whose dependence on the number of states would be better than $O(N \ln K)$. Seldin et al. (2011a) also demonstrate that empirically it takes a large number of rounds until the asymptotic advantage of UCB over EXP3 translates into a real advantage in practice.

It is not trivial to minimize (4) with respect to $\gamma_t$ analytically. Generally, higher values of $\gamma_t$ decrease the second term of the bound, but also lead to more concentrated policies (conditional distributions) $\rho_t^{exp}(a|s)$ and thus higher mutual information values $I_{\rho_t^{exp}}(S; A)$. A simple way to address this trade-off is to set $\gamma_t$ such that the contribution of the second term is as close to the contribution of the first term as possible. This can be approximated by taking the value of mutual information from the previous round (or approximation of the value of mutual information from the previous round). More details on parameter setting for the algorithm are provided in the supplementary material.

**Comments on Algorithm 1.** By regret decomposition (5) and Theorem 2, regret at round $t + 1$ is minimized by a policy $\rho_t(a|s)$ that minimizes a certain trade-off between the mutual information $I_\rho(S; A)$ and the empirical regret $\hat{R}_t(\rho)$. This trade-off is analogical to rate-distortion trade-off in information theory (Cover and Thomas, 1991). Minimization of rate-distortion trade-off is achieved by iterative updates of the following form, which are known as Blahut-Arimoto (BA) algorithm:

$$\rho_t^{BA}(a|s) = \frac{\rho_t^{BA}(a)e^{\gamma_t \hat{R}_t(a,s)}}{\sum_a \rho_t^{BA}(a)e^{\gamma_t \hat{R}_t(a,s)}}, \qquad \rho_t^{BA}(a) = \frac{1}{N}\sum_s \rho_t^{BA}(a|s).$$

Running a similar type of iterations in our case would be prohibitively expensive, since they require iteration over all states $s \in \mathcal{S}$ at each round of the game. We approximate these iterations by approximating the marginal distribution over the actions by a running average:

$$\tilde{\rho}_{t+1}^{exp}(a) = \frac{N-1}{N}\tilde{\rho}_t^{exp}(a) + \frac{1}{N}\tilde{\rho}_t^{exp}(a|S_t). \tag{7}$$

Since $\rho_t^{exp}(a|s)$ is bounded from zero by a decreasing sequence $\varepsilon_{t+1}$, the same automatically holds for $\tilde{\rho}_{t+1}^{exp}(a)$ (meaning that in Theorem 1 $\epsilon_t = \varepsilon_t$). Note that Theorem 1 holds for any choice of $\rho_t(a)$, including (7).

We point out an interesting fact: $\rho_t^{exp}(a)$ propagates information between different states, but Theorem 1 also holds for the uniform distribution $\rho(a) = \frac{1}{K}$, which corresponds to application of EXP3 algorithm in each state independently. If these independent multiarmed bandits independently converge to similar strategies, we still get a tighter regret bound. This happens because the corresponding subspace of the hypothesis space is significantly smaller than the total hypothesis space, which enables us to put a higher prior on it (Seldin and Tishby, 2010). Nevertheless, propagation of information between states via the distribution $\rho_t^{exp}(a)$ helps to achieve even faster convergence of the regret, as we can see from the experiments in the next section.

**Comparison with state-of-the-art.** We are not aware of algorithms for stochastic multiarmed bandits with side information. The best known to us algorithm for adversarial multiarmed bandits with side information is EXP4.P by Beygelzimer et al. (2011). EXP4.P has $O(\sqrt{Kt \ln |\mathcal{H}|})$ regret and $O(K|\mathcal{H}|)$ complexity per game round. In our case $|\mathcal{H}| = K^N$, which means that EXP4.P would have $O(\sqrt{KtN \ln K})$ regret and $O(K^{N+1})$ computational complexity. For hard problems, where all side information has to be used, our regret bound is inferior to the regret bound of Beygelzimer et al. (2011) due to $O(t^{2/3})$ dependence on the number of game rounds. However, we believe that this can be improved by a more careful analysis of the existing algorithm. For simple problems the dependence of our regret bound on the number of states is significantly better, up to the point that when the side information is irrelevant for the task we can get $O(\sqrt{K \ln N})$ dependence on the number of states versus $O(\sqrt{N \ln K})$ in EXP4.P. For $N \gg K$ this leads to tighter regret bounds for small $t$ even despite the "incorrect" dependence on $t$ of our bound, and if we improve the analysis it will lead to tighter regret bounds for all $t$. As we said it already, our algorithm is able to filter relevant information from large amounts of side information automatically, whereas in EXP4.P the usage of side information has to be restricted externally through the construction of the hypothesis class.

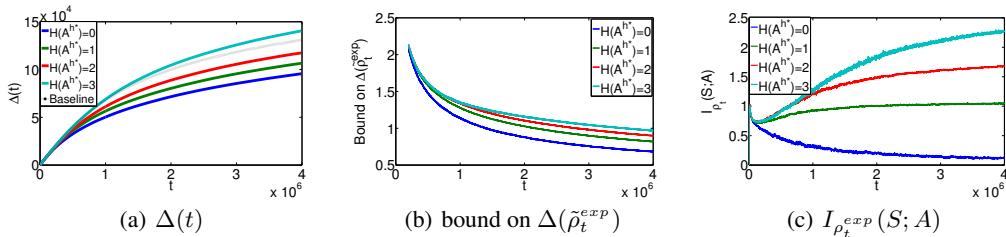

(a) $\Delta(t)$  (b) bound on $\Delta(\tilde{\rho}_t^{exp})$  (c) $I_{\rho_t^{exp}}(S; A)$

**Figure 1: Behavior of: (a) cumulative regret $\Delta(t)$, (b) bound on instantaneous regret $\Delta(\tilde{\rho}_t^{exp})$, and (c) the approximation of mutual information $I_{\rho_t^{exp}}(S; A)$.** "Baseline" in the first graph corresponds to playing $N$ independent multiarmed bandits, one in each state. Each line in the graphs corresponds to an average over 10 repetitions of the experiment.

The second important advantage of our algorithm is the exponential improvement of computational complexity. This is achieved by switching from the space of experts to the state-action space in all our calculations.

## 4 Experiments

We present an experiment on synthetic data that illustrates our results. We take $N = 100$, $K = 20$, a uniform distribution over states ($p(s) = 0.01$), and consider four settings, with $H(A^{h^*}) = \ln(1) = 0$, $H(A^{h^*}) = \ln(3) \approx 1$, $H(A^{h^*}) = \ln(7) \approx 2$, and $H(A^{h^*}) = \ln(20) \approx 3$, respectively. In the first case, the same action is the best in all states (and hence $H(A^{h^*}) = 0$ for the optimal hypothesis $h^*$). In the second case, for the first 33 states the best action is number 1, for the next 33 states the best action is number 2, and for the rest third of the states the best action is number 3 (thus, depending on the state, one of the three actions is the "best" and $H(A^{h^*}) = \ln(3)$). In the third case, there are seven groups of 14 states and each group has its own best action. In the last case, there are 20 groups of 5 states and each of $K = 20$ actions is the best in exactly one of the 20 groups. For all states, the reward of the best action in a state has Bernoulli distribution with bias 0.6 and the rewards of all other actions in that state have Bernoulli distribution with bias 0.5. We run the experiment for $T = 4,000,000$ rounds and calculate the cumulative regret $\Delta(t) = \sum_{\tau=1}^{t} \Delta(\tilde{\rho}_\tau^{exp})$ and instantaneous regret bound given in (4). For computational efficiency, the mutual information $I_{\rho_t^{exp}}(S; A)$ is approximated by a running average (see supplementary material for details).

As we can see from the graphs (see Figure 1), the algorithm exhibits sublinear cumulative regret (put attention to the axes' scales). Furthermore, for simple problems (with small $H(A^{h^*})$) the regret grows slower than for complex problems. "Baseline" in Figure 1.a shows the performance of an algorithm with the same parameter values that runs $N$ multiarmed bandits, one in each state independently of other states. We see that for all problems except the hardest one our algorithm performs better than the baseline and for the hardest problem it performs almost as good as the baseline. The regret bound in Figure 1.b provides meaningful values for the simplest problem after 1 million rounds (which is on average 500 samples per state-action pair) and after 4 million rounds for all the problems (the graph starts at $t = 10,000$). Our estimates of the mutual information $I_{\rho_t^{exp}}(S; A)$ reflect $H(A^{h^*})$ for the corresponding problems (for $H(A^{h^*}) = 0$ it converges to zero, for $H(A^{h^*}) \approx 1$ it is approximately one, etc.).

## 5 Proof of Theorem 2

The proof of Theorem 2 is based on PAC-Bayes-Bernstein inequality for martingales (Seldin et al., 2011b). Let $\mathrm{KL}(\rho\|\mu)$ denote the KL-divergence between two distributions (Cover and Thomas, 1991). Let $\{Z_1(h), \ldots, Z_n(h) : h \in \mathcal{H}\}$ be martingale difference sequences indexed by $h$ with respect to the filtration $\sigma(\mathcal{U}_1), \ldots, \sigma(\mathcal{U}_n)$, where $\mathcal{U}_i = \{Z_1(h), \ldots, Z_i(h) : h \in \mathcal{H}\}$ is the subset of martingale difference variables up to index $i$ and $\sigma(\mathcal{U}_i)$ is the $\sigma$-algebra generated by $\mathcal{U}_i$. This means that $\mathbb{E}[Z_i(h)|\sigma(\mathcal{U}_{i-1})] = 0$, where $Z_i(h)$ may depend on $Z_j(h')$ for all $j < i$ and $h' \in \mathcal{H}$. There might also be interdependence between $\{Z_i(h) : h \in \mathcal{H}\}$. Let $\hat{M}_i(h) = \sum_{j=1}^{i} Z_j(h)$ be

the corresponding martingales. Let $V_i(h) = \sum_{j=1}^{i} \mathbb{E}[Z_j(h)^2|\sigma(\mathcal{U}_{j-1})]$ be cumulative variances of the martingales $\hat{M}_i(h)$. For a distribution $\rho$ over $\mathcal{H}$ define $\hat{M}_i(\rho) = \mathbb{E}_{\rho(h)}[\hat{M}_i(h)]$ and $V_t(\rho) = \mathbb{E}_{\rho(h)}[V_t(h)]$ as weighted averages of the martingales and their cumulative variances according to a distribution $\rho$.

**Theorem 3** (PAC-Bayes-Bernstein Inequality). *Assume that $|Z_i(h)| \leq b$ for all $h$ with probability 1. Fix a prior distribution $\mu$ over $\mathcal{H}$. Pick an arbitrary number $c > 1$. Then with probability greater than $1 - \delta$ over $\mathcal{U}_n$, simultaneously for all distributions $\rho$ over $\mathcal{H}$ that satisfy*

$$\sqrt{\frac{\mathrm{KL}(\rho\|\mu) + \ln\frac{2m}{\delta}}{(e-2)V_n(\rho)}} \leq \frac{1}{cb}$$

*we have*

$$|\hat{M}_n(\rho)| \leq (1+c)\sqrt{(e-2)V_n(\rho)\left(\mathrm{KL}(\rho\|\mu) + \ln\frac{2m}{\delta}\right)},$$

*where $m = \ln\left(\sqrt{\frac{(e-2)n}{\ln\frac{2}{\delta}}}\right) / \ln(c)$, and for all other $\rho$*

$$|\hat{M}_n(\rho)| \leq 2b\left(KL(\rho\|\mu) + \ln\frac{2m}{\delta}\right).$$

Note that $M_t(h) = t(\Delta(h) - \hat{\Delta}_t(h))$ are martingales and their cumulative variances are $V_t(h) = \sum_{\tau=1}^{t} \mathbb{E}\big[\big([R_\tau^{h^*(S_\tau),S_\tau} - R_\tau^{h(S_\tau),S_\tau}] - [R(h^*) - R(h)]\big)^2\big|\mathcal{T}_{\tau-1}\big]$. In order to apply Theorem 3 we have to derive an upper bound on $V_t(\rho_t^{exp})$,[1] a prior $\mu(h)$ over $\mathcal{H}$, and calculate (or upper bound) the KL-divergence $\mathrm{KL}(\rho_t^{exp}\|\mu)$. This is done in the following three lemmas.

**Lemma 3.** *If $\{\varepsilon_1, \varepsilon_2, \dots\}$ is a decreasing sequence, such that $\varepsilon_t \leq \min_{a,s} \pi_t(a|s)$, then for all $h$:*

$$V_t(h) \leq \frac{2t}{\varepsilon_t}.$$

The proof of the lemma is provided in the supplementary material. Lemma 3 provides an immediate, but crude, uniform upper bound on $V_t(h)$, which yields $V_t(\rho_t^{exp}) \leq \frac{2t}{\varepsilon_t}$. Since our algorithm concentrates on $h$-s with small $\Delta(h)$, which, in turn, concentrate on the best action in each state, the variance $V_t(h)$ for the corresponding $h$-s is expected to be of the order of $2Kt$ and not $\frac{2t}{\varepsilon_t}$. However, we were not able to prove yet that the probability $\rho_t^{exp}(h)$ of the remaining hypotheses (those with large $\Delta(h)$) gets sufficiently small (of order $K\varepsilon_t$), so that the weighted cumulative variance would be of order $2Kt$. Nevertheless, this seems to hold in practice starting from relatively small values of $t$ (Seldin et al., 2011a). Improving the upper bound on $V_t(\rho_t^{exp})$ will improve the regret bound, but for the moment we present the regret bound based on the crude upper bound $V_t(\rho_t^{exp}) \leq \frac{2t}{\varepsilon_t}$.

The remaining two lemmas, which define a prior $\mu$ over $\mathcal{H}$ and bound $\mathrm{KL}(\rho\|\mu)$, are due to Seldin and Tishby (2010).

**Lemma 4.** *It is possible to define a distribution $\mu$ over $\mathcal{H}$ that satisfies*:

$$\mu(h) \geq e^{-NH(A^h) - K\ln N - K\ln K}. \tag{8}$$

**Lemma 5.** *For the distribution $\mu$ that satisfies (8) and any distribution $\rho(a|s)$*:

$$\mathrm{KL}(\rho\|\mu) \leq NI_\rho(S;A) + K\ln N + K\ln K.$$

Substitution of the upper bounds on $V_t(\rho_t^{exp})$ and $\mathrm{KL}(\rho_t^{exp}\|\mu)$ into Theorem 3 yields Theorem 2.

## 6   Discussion

We presented PAC-Bayesian analysis of stochastic multiarmed bandits with side information. Our analysis provides data-dependent algorithm and data-dependent regret analysis for this problem. The selection of task-relevant side information is delegated from the user to the algorithm. We also provide a general framework for deriving data-dependent algorithms and analyses for many other stochastic problems with limited feedback. The analysis of the variance of our algorithm still waits to be improved and will be addressed in future work.

## Acknowledgments

We would like to thank all the people with whom we discussed this work and, in particular, Nicolò Cesa-Bianchi, Gábor Bartók, Elad Hazan, Csaba Szepesvári, Miroslav Dudík, Robert Shapire, John Langford, and the anonymous reviewers, whose comments helped us to improve the final version of this manuscript. This work was supported in part by the IST Programme of the European Community, under the PASCAL2 Network of Excellence, IST-2007-216886, and by the European Community's Seventh Framework Programme (FP7/2007-2013), under grant agreement $N^o$231495. This publication only reflects the authors' views.

## Footnotes

[1]Seldin et al. (2011b) show that $V_n(\rho)$ can be replaced by an upper bound everywhere in Theorem 3.

## References

Peter Auer, Nicolò Cesa-Bianchi, and Paul Fischer. Finite-time analysis of the multiarmed bandit problem. *Machine Learning*, 47, 2002a.

Peter Auer, Nicolò Cesa-Bianchi, Yoav Freund, and Robert E. Schapire. The nonstochastic multiarmed bandit problem. *SIAM Journal of Computing*, 32(1), 2002b.

Arindam Banerjee. On Bayesian bounds. In *Proceedings of the International Conference on Machine Learning (ICML)*, 2006.

Alina Beygelzimer, John Langford, Lihong Li, Lev Reyzin, and Robert Schapire. Contextual bandit algorithms with supervised learning guarantees. In *Proceedings on the International Conference on Artificial Intelligence and Statistics (AISTATS)*, 2011.

Thomas M. Cover and Joy A. Thomas. *Elements of Information Theory*. John Wiley & Sons, 1991.

Leslie Pack Kaelbling. Associative reinforcement learning: Functions in $k$-DNF. *Machine Learning*, 15, 1994.

John Langford and Tong Zhang. The epoch-greedy algorithm for contextual multi-armed bandits. In *Advances in Neural Information Processing Systems (NIPS)*, 2007.

David McAllester. Some PAC-Bayesian theorems. In *Proceedings of the International Conference on Computational Learning Theory (COLT)*, 1998.

Matthias Seeger. PAC-Bayesian generalization error bounds for Gaussian process classification. *Journal of Machine Learning Research*, 2002.

Yevgeny Seldin and Naftali Tishby. PAC-Bayesian analysis of co-clustering and beyond. *Journal of Machine Learning Research*, 11, 2010.

Yevgeny Seldin, Nicolò Cesa-Bianchi, Peter Auer, François Laviolette, and John Shawe-Taylor. PAC-Bayes-Bernstein inequality for martingales and its application to multiarmed bandits. 2011a. In review. Preprint available at http://arxiv.org/abs/1110.6755.

Yevgeny Seldin, François Laviolette, Nicolò Cesa-Bianchi, John Shawe-Taylor, and Peter Auer. PAC-Bayesian inequalities for martingales. 2011b. In review. Preprint available at http://arxiv.org/abs/1110.6886.

John Shawe-Taylor and Robert C. Williamson. A PAC analysis of a Bayesian estimator. In *Proceedings of the International Conference on Computational Learning Theory (COLT)*, 1997.

John Shawe-Taylor, Peter L. Bartlett, Robert C. Williamson, and Martin Anthony. Structural risk minimization over data-dependent hierarchies. *IEEE Transactions on Information Theory*, 44(5), 1998.

Alexander L. Strehl, Chris Mesterharm, Michael L. Littman, and Haym Hirsh. Experience-efficient learning in associative bandit problems. In *Proceedings of the International Conference on Machine Learning (ICML)*, 2006.

Richard S. Sutton and Andrew G. Barto. *Reinforcement Learning: An Introduction*. MIT Press, 1998.

Naftali Tishby and Daniel Polani. Information theory of decisions and actions. In Vassilis Cutsuridis, Amir Hussain, John G. Taylor, and Daniel Polani, editors, *Perception-Reason-Action Cycle: Models, Algorithms and Systems*. Springer, 2010.

Naftali Tishby, Fernando Pereira, and William Bialek. The information bottleneck method. In *Allerton Conference on Communication, Control and Computation*, 1999.

Leslie G. Valiant. A theory of the learnable. *Communications of the Association for Computing Machinery*, 27 (11), 1984.

